# Near-Maximum Entropy Models for Binary Neural Representations of Natural Images

**Matthias Bethge and Philipp Berens**
Max Planck Institute for Biological Cybernetics
Spemannstrasse 41, 72076, Tübingen, Germany
`mbethge,berens@tuebingen.mpg.de`

## Abstract

Maximum entropy analysis of binary variables provides an elegant way for studying the role of pairwise correlations in neural populations. Unfortunately, these approaches suffer from their poor scalability to high dimensions. In sensory coding, however, high-dimensional data is ubiquitous. Here, we introduce a new approach using a near-maximum entropy model, that makes this type of analysis feasible for very high-dimensional data—the model parameters can be derived in closed form and sampling is easy. Therefore, our *NearMaxEnt* approach can serve as a tool for testing predictions from a pairwise maximum entropy model not only for low-dimensional marginals, but also for high dimensional measurements of more than thousand units. We demonstrate its usefulness by studying natural images with dichotomized pixel intensities. Our results indicate that the statistics of such higher-dimensional measurements exhibit additional structure that are not predicted by pairwise correlations, despite the fact that pairwise correlations explain the lower-dimensional marginal statistics surprisingly well up to the limit of dimensionality where estimation of the full joint distribution is feasible.

## 1 Introduction

A core issue in sensory coding is to seek out and model statistical regularities in high-dimensional data. In particular, motivated by developments in information theory, it has been hypothesized that modeling these regularities by means of redundancy reduction constitutes an important goal of early visual processing [2]. Recent studies conjectured that the binary spike responses of retinal ganglion cells may be characterized completely in terms of second-order correlations when using a maximum entropy approach [13, 12]. In light of what we know about the statistics of the visual input, however, this would be very surprising: Natural images are known to exhibit complex higher-order correlations which are extremely difficult to model yet being perceptually relevant. Thus, if we assume that retinal ganglion cells do not discard the information underlying these higher-order correlations altogether, it would be a very difficult signal processing task to remove all of those already within the retinal network.

Oftentimes, neurons involved in early visual processing are modeled as rather simple computational units akin to generalized linear models, where a linear filter is followed by a point-wise nonlinearity. For such simple neuron models, the possibility of removing higher-order correlations present in the input is very limited [3].

Here, we study the role of second-order correlations in the multivariate binary output statistics of such linear-nonlinear model neurons with a threshold nonlinearity responding to natural images. That is, each unit can be described by an affine transformation $z_k = \mathbf{w}_k^T \mathbf{x} + \vartheta$ followed by a point-wise *signum* function $s_k = \mathrm{sgn}(z_k)$. Our interest in this model is twofold: (A) It can be regarded a parsimonious model for the analysis of population codes of natural images for which the

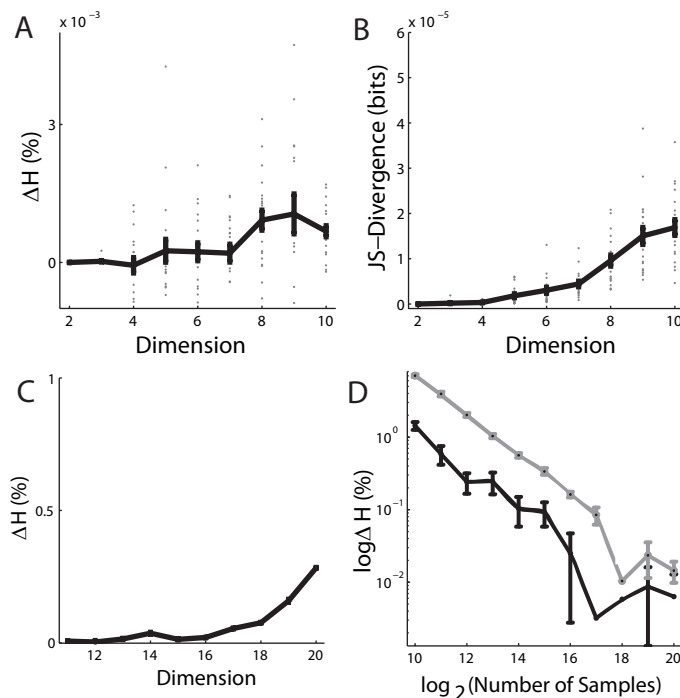

Figure 1: Similarity between the Ising and the DG model. **A+C:** Entropy difference $\Delta H$ between the Ising model and the Dichotomized Gaussian distribution as a function of dimensionality. **A:** Up to 10 dimensions we can compute $H_{DG}$ directly by evaluating Eq. 6. Gray dots correspond to different sets of parameters. For $m \geq 4$, the relatively large scatter and the existence of negative values is due to the limited numerical precision of the Monte-Carlo integration. Errorbars show standard error of the mean. **B.** JS-divergence $D_{JS}$ between $P_I$ and $P_{DG}$. **C.** $\Delta H$ as above, for higher dimensions. Up to 20 dimensions $\Delta H$ remains very small. The increase for $m \to 20$ is most likely due to undersampling of the distributions. **D.** $\Delta H$ as function of sample size used to estimate $H_{DG}$, at seven (black) and ten (grey) dimensions (note log scale on both axes). $\Delta H$ decreases with a power law with increasing sample sizes.

computational power and the bandwidth of each unit is limited. (B) The same model can also be used more generally to fit multivariate binary data with given pairwise correlations, if **x** is drawn from a Gaussian distribution. In particular, we will show that the resulting distribution closely resembles the binary maximum entropy models known as Ising models or Boltzmann machines which have recently become popular for the analysis of spike train recordings from retinal ganglion cell responses [13, 12].

Motivated by the analysis in [12, 13] and the discussion in [10] we are interested at a more general level in the following questions: are pairwise interactions enough for understanding the statistical regularities in high-dimensional natural data (given that they provide a good fit in the low-dimensional case)? If we suppose that pairwise interactions are enough, what can we say about the amount of redundancies in high-dimensional data? In comparison with neural spike data, natural images provide two advantages for studying these questions: 1) It is much easier to obtain large amounts of data with millions of samples which are less prone to nonstationarities. 2) Often differences in the higher-order statistics such as between pink noise and natural images can be recognized by eye.

## 2 Second order models for binary variables

In order to study whether pairwise interactions are enough to determine the statistical regularities in high-dimensional data, it is necessary to be able to compute the maximum entropy distribution for large number of dimensions $N$. Given a set of measured statistics, maximum entropy models yield a full probability distribution that is consistent with these constraints but does not impose any

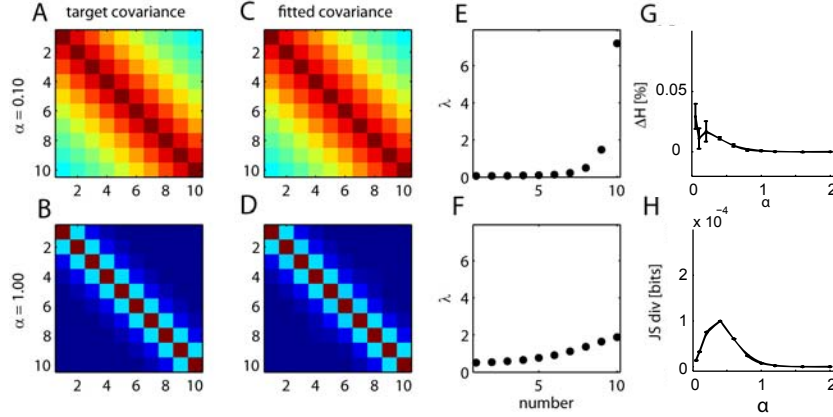

Figure 2: Examples of covariance matrices (**A+B.**) and their learned approximations (**C+D**) at $m = 10$ for clarity. $\alpha$ is the parameter controlling the steepness of correlation decrease. **E+F.** Eigenvalue spectra of both matrices. **G.** Entropy difference $\Delta H$ and **H.** JS-divergence between the distribution of samples obtained from the two models at $m = 7$.

additional structure on the distribution [7]. For binary data with given mean activations $\mu_i = \langle s_i \rangle$ and correlations between neurons $\Sigma_{ij} = \langle s_i s_j \rangle - \langle s_i \rangle \langle s_j \rangle$, one obtains a *quadratic exponential* probability mass function known as the *Ising model* in physics or as the *Boltzmann machine* in machine learning.

Currently all methods used to determine the parameters of such binary maximum entropy models suffer from the same drawback: since the parameters do not correspond directly to any of the measured statistics, they have to be inferred (or 'learned') from data. In high dimensions though, this poses a difficult computational problem. Therefore the characterization of complete neural circuits with possibly hundreds of neurons is still out of reach, even though analysis was recently extended to up to forty neurons [14].

To make the maximum entropy approach feasible in high dimensions, we propose a new strategy: Sampling from a 'near-maximum' entropy model that does not require any complicated learning of parameters. In order to justify this approach, we verify empirically that the entropy of the full probability distributions obtained with the near-maximum entropy model are indistinguishable from those obtained with classical methods such as Gibbs sampling for up to 20 dimensions.

## 2.1 Boltzmann machine learning

For a binary vector of neural activities $\mathbf{s} \in \{-1, 1\}^m$ and specified $\mu_i$ and $\Sigma_{ij}$ the Ising model takes the form

$$P_I(\mathbf{s}) = \frac{1}{Z} \exp \left[ \sum_{i=1}^{m} h_i s_i + \frac{1}{2} \sum_{i \neq j} J_{ij} s_i s_j \right], \tag{1}$$

where the local fields $h_i$ and the couplings $J_{ij}$ have to be chosen such that $\langle s_i \rangle = \mu_i$ and $\langle s_i s_j \rangle - \langle s_i \rangle \langle s_j \rangle = \Sigma_{ij}$. Unfortunately, finding the correct parameters turns out to be a difficult problem which cannot be solved in closed form.

Therefore, one has to resort to an optimization approach to learn the model parameters $h_i$ and $J_{ij}$ from data. This problem is called Boltzmann machine learning and is based on maximization of the log-likelihood $L = \ln P_I(\{\mathbf{s}_i\}_{i=1}^{N} | h, J)$ [1] where $N$ is the number of samples. The gradient of the likelihood can be computed in terms of the empirical covariance and the covariance of $s_i$ and $s_j$ as produced by the current model:

$$\frac{\partial L}{\partial J_{ij}} = \langle s_i s_j \rangle_{\text{Data}} - \langle s_i s_j \rangle_{\text{Model}} \tag{2}$$

The second term on the right hand side is difficult to compute, as it requires sampling from the model. Since the partition function $Z$ in Eq. (1) is not available in closed form, Monte-Carlo methods such

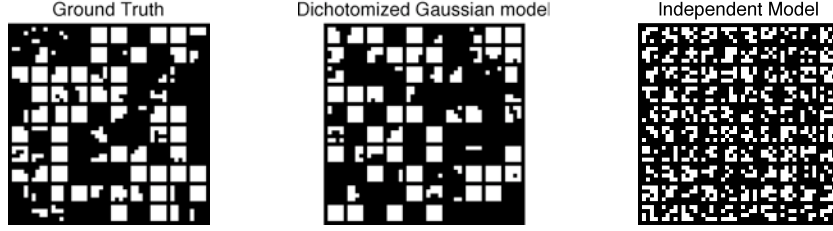

Figure 3: Random samples of dichotomized 4x4 patches from the van Hateren image data base (left) and from the corresponding dichotomized Gaussian distribution with equal covariance matrix (middle). It is not possible to see any systematic difference between the samples from the two distributions. For comparison, this is not so for the sample from the independent model (right).

as Gibbs sampling are employed [9] in order to approximate the required model average. This is computationally demanding as sampling is necessary for each individual update. While efficient sampling algorithms exist for special cases [6], it still remains a hard and time consuming problem in the general case. Additionally, most sampling algorithms do not come with guarantees for the quality of the approximation of the required average. In conclusion, parameter fitting of the Ising model is slow and oftentimes painstaking, especially in high dimensions.

## 2.2 Modeling with the dichotomized Gaussian

Here we explore an intriguing alternative to the Monte-Carlo approach: We replace the Ising model by a 'near-maximum' entropy model, for which both parameter computation and sampling is easy. A very convenient, but in this context rarely recognized, candidate model is the dichotomized Gaussian distribution (DG) [11, 5, 4]. It is obtained by supposing that the observed binary vector $\mathbf{s}$ is generated from a hidden Gaussian variable

$$\mathbf{z} \sim \mathcal{N}(\gamma, \Lambda) \ , \quad s_i = \text{sgn}(z_i). \tag{3}$$

Without loss of generality, we can assume unit variances for the Gaussian, i.e. $\Lambda_{ii} = 1$, the mean $\mu$ and the covariance matrix $\Sigma$ of $\mathbf{s}$ are given by

$$\mu_i = 2\Phi(\gamma_i) - 1 \ , \quad \Sigma_{ii} = 4\Phi(\gamma_i)\Phi(-\gamma_i) \ , \quad \Sigma_{ij} = 4\Psi(\gamma_i, \gamma_j, \Lambda_{ij}) \text{ for } i \neq j \tag{4}$$

where $\Psi(x, y, \lambda) = \Phi_2(x, y, \lambda) - \Phi(x)\Phi(y)$ . Here $\Phi$ is the univariate standardized cumulative Gaussian distribution and $\Phi_2$ its bivariate counterpart. While the computation of the model parameters was hard for the Ising model, these equations can be easily inverted to find the parameters of the hidden Gaussian distribution:

$$\gamma_i = \Phi^{-1}\left(\frac{\mu_i + 1}{2}\right) \tag{5}$$

Determining $\Lambda_{ij}$ generally requires to find a suitable value such that $\Sigma_{ij} - 4\Psi(\gamma_i, \gamma_j, \Lambda_{ij}) = 0$. This can be efficently solved by numerical computations, since the function is monotonic in $\Lambda_{ij}$ and has a unique zero crossing. We obtain an especially easy case, when $\gamma_i = \gamma_j = 0$, as then $\Lambda_{ij} = \sin\left(\frac{\pi}{2}\Sigma_{ij}\right)$.

It is also possible to evaluate the probability mass function of the DG model by numerical integration,

$$P_{DG}(\mathbf{s}) = \frac{1}{(2\pi)^{N/2}|\Lambda|^{1/2}} \int_{a_1}^{b_1} \ldots \int_{a_m}^{b_m} \exp\left(-(\mathbf{s} - \gamma)^T \Lambda^{-1}(\mathbf{s} - \gamma)\right), \tag{6}$$

where the integration limits are chosen as $a_i = 0$ and $b_i = \infty$, if $s_i = 1$, and $a_i = -\infty$ and $b_i = 0$, otherwise.

In summary, the proposed model has two advantages over the traditional Ising model: (1) Sampling is easy, and (2) finding the model parameters is easy too.

# 3 Near-maximum entropy behavior of the dichotomized Gaussian distribution

In the previous section we introduced the dichotomized Gaussian distribution. Our conjecture is that in many cases it can serve as a convenient approximation to the Ising model. Now, we investigate how good this approximation is. For a wide range of interaction terms and mean activations we verify that the DG model closely resembles the Ising model. In particular we show that the entropy of the DG distribution is not smaller than the entropy of the Ising model even at rather high dimensions.

## 3.1 Random Connectivity

We created randomly connected networks of varying size $m$, where mean activations $h_i$ and interactions terms $J_{ij}$ were drawn from $\mathcal{N}(0, 0.4)$. First, we compared the entropy $H_I = -\sum_{\mathbf{s}} P_I(\mathbf{s}) \log_2 P_I(\mathbf{s})$ of the thus specified Ising model obtained by evaluating Eq. 1 with the entropy of the DG distribution $H_{DG}$ computed by numerical integration[1] from Eq. 6 (twenty parameter sets). The entropy difference $\Delta H = H_I - H_{DG}$ was smaller than 0.002 percent of $H_I$ (Fig. 1 A, note scale) and probably within the range of the numerical integration accuracy. In addition, we computed the *Jensen-Shannon divergence* $D_{JS}[P_I \| P_{DG}] = \frac{1}{2} (D_{KL}[P_I \| M] + D_{KL}[P_{DG} \| M])$, where $M = \frac{1}{2}(P_I + P_{DG})$ [8]. We find that $D_{JS}[P_I \| P_{DG}]$ is extremly small up to 10 dimensions (Fig. 1 B). Therefore, the distributions seem to be not only close in their respective entropy, but also to have a very similar structure.

Next, we extended this analysis to networks of larger size and repeated the same analysis for up to twenty dimensions. Since the integration in Eq. 6 becomes too time-consuming for $m \to 20$ due to the large number of states, we used a histogram based estimate of $P_{DG}$ (using $3 \cdot 10^6$ samples for $m < 15$ and $15 \cdot 10^6$ samples for $m \geq 15$). The estimate of $\Delta H$ is still very small at high dimensions (Fig. 1 C, below 0.5%). We also computed $D_{JS}$, which scaled similarly to $\Delta H$ (data not shown).

In Fig. 1 C, $\Delta H$ seems to increase with dimensionality. Therefore, we investigated how the estimate of $\Delta H$ is influenced by the number of samples used. We computed both quantities for varying numbers of samples from the DG distribution (for $m = 7, 10$). As $\Delta H$ decreases according to a power law with increasing $m$, the rise of $\Delta H$ observed in Fig. 1 C is most likely due to undersampling of the distribution.

## 3.2 Specified covariance structure

To explore the relationship between the two techniques more systematically, we generated covariance matrices with varying eigenvalue spectra. We used a parametric Toeplitz form, where the $n$th diagonal is set to a constant value $\exp(-\alpha \cdot n)$ (Fig. 2A and B, $m = 7, 10$). We varied the decay parameter $\alpha$, which led to a widely varying covariance structure (For eigenvalue spectra, see Fig. 2E and F). We fit the Ising models using the Boltzmann machine gradient descent procedure. The covariance matrix of the samples drawn from the Ising model resembles the original very closely (Fig. 2C and D). We also computed the entropy of the DG model using the desired covariance structure. We estimated $\Delta H$ and $D_{JS}[P_G \| P_{DG}]$ averaged over 10 trials with $10^5$ samples obtained by Gibbs sampling from the Ising model. $\Delta H$ is very close to zero (Fig. 2G, $m = 7$) except for small $\alpha$s and never exceeded 0.05%. Moreover, the structure of both distributions seems to be very similar as well (Fig. 2H, $m = 7$). At $m = 10$, both quantities scaled qualitatively similair (data not shown). We also repeated this analysis using equations 1 and 6 as before, which lead to similar results (data not shown).

Our experiments demonstrate clearly that the dichotomized Gaussian distribution constitutes a good approximation to the quadratic exponential distribution for a large parameter range. In the following section, we will exploit the similarity between the two models to study how the role of second-order correlations may change between low-dimensional and high-dimensional statistics in case of natural images.

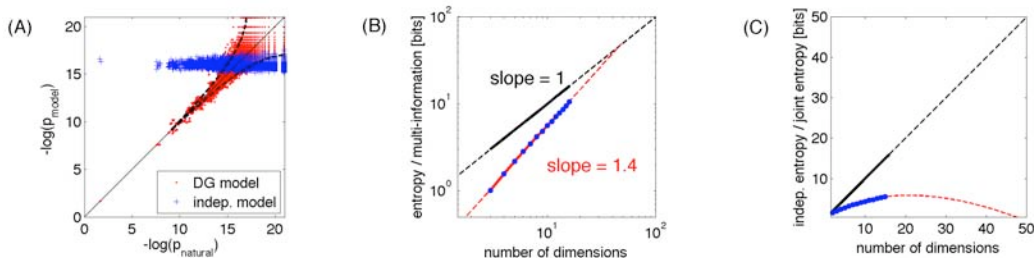

Figure 4: **A:** Negative log probabilities of the DG model are plotted against ground truth (red dots). Identical distributions fall on the diagonal. Data points outside the area enclosed by the dashed lines indicate significant differences between the model and ground truth. The DG model matches the true distribution very well. For comparison the independent model is shown as well (blue crosses). **B:** The multi-information of the true distribution (blue dots) accurately agrees with the multi-information of the DG model (red line). Similar to the analysis in [12], we observe a power law behavior of the entropy of the independent model (black solid line) and the mutli-information. Linear extrapolation (in the log-log plot) to higher dimensions is indicated by dashed lines. **C:** Different way of presentation of the same data as in B: the joint entropy $H = H_{indep} - I$ (blue dots) is plotted instead of $I$ and the axis are in linear scale. The dashed red line represents the same extrapolation as in B.

# 4   Natural images: Second order and beyond

We now investigate to which extent the statistics of natural images with dichotomized pixel intensities can be characterized by pairwise correlations only. In particular, we would like to know how the role of pairwise correlations opposed to higher-order correlations changes depending on the dimensionality. Thanks to the DG model introduced above, we are in the position to study the effect of pairwise correlations for high-dimensional binary random variables ($N \approx 1000$ or even larger).

We use the van Hateren image database in log-intensity scale, from which we sample small image patches at random positions. The threshold for the dichotomization is set to the median of pixel intensities. That is, each binary variable encodes whether the corresponding pixel intensity is above or below the median over the ensemble. Up to patch sizes of $4 \times 4$ pixel, the true joint statistics can be assessed using nonparametric histogram methods. Before we present quantitative comparisons, it is instructive to look at random samples from the true distribution (Fig. 3, left), from the DG model with same mean and covariance (Fig. 3, middle), and from the corresponding independent model (Fig. 3, right). By visual inspection, it seems that the DG model fits the true distribution well.

In order to quantify *how* well the DG model matches the true distribution, we draw two independent sets of samples from each ($N = 2 \cdot 10^6$ for each set) and generate a scatter plot as shown in Fig. 4 A for $4 \times 4$ image patches. Each dot corresponds to one of the $2^{16} = 65536$ possible different binary patterns. The relative frequencies of these patterns according to the DG model (red dots) and according to the independent model (blue dots) are plotted against the relative frequencies obtained from the natural image patches. The solid diagonal line corresponds to a perfect match between model and ground truth. The dashed lines enclose the regions within which deviations are to be expected due to the finite sampling size. Since most of the red dots fall within this region, the DG model fits the data distribution very well.

We also systematically evaluated the JS-divergence and the multi-information $I[\mathbf{S}] = \sum_k H[S_k] - H[\mathbf{S}]$ as a function of dimensionality. That is, we started with the bivariate marginal distribution of two randomly selected pixels. Then we incrementally added more pixels of random location until the random vector contains all the 16 pixels of the $4 \times 4$ image patches. Independent of the dimension, the JS-divergence between the DG model and the true distribution is smaller than 0.015 bits. For comparison, the JS-divergence between the independent model and the true distribution increases with dimensionality from roughly 0.2 bits in the case of two pixels up to 0.839 bits in the case of 16 pixels. For two independent sets of samples both drawn from natural image data the JS-divergence ranges between 0.006 and 0.007 bits for $4 \times 4$ patches setting the gold standard for the minimal possible JS-divergence one could achieve with any model due to finite sampling size.

Carrying out the same type of analysis as in [12], we make qualitatively the same observations as it was reported there: as shown above, we find a quite accurate match between the two distributions.

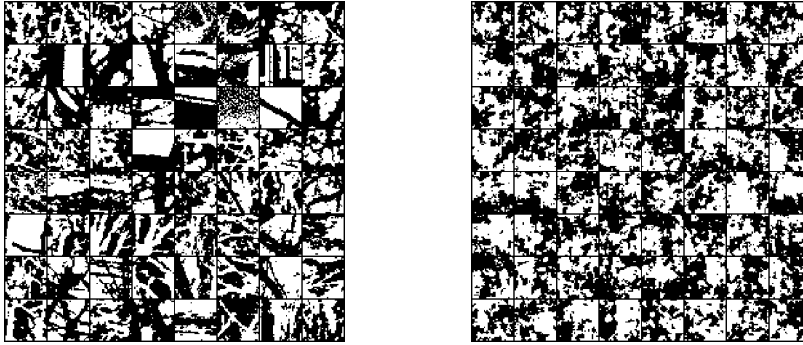

Figure 5: Random samples of dichotomized 32x32 patches from the van Hateren image data base (left) and from the corresponding dichotomized Gaussian distribution with equal covariance matrix (right). For the latter, the percept of typical objects is missing due to the ignorance of higher-order correlations. This striking difference is not obvious, however, at the level of 4x4 patches, for which we found an excellent match of the dichotomized Gaussian to the ensemble of natural images.

Furthermore, the multi-information of the DG model (red solid line) and of the true distribution (blue dots) increases linearly on a loglog-scale with the number of dimensions (Fig. 4 B). Both findings can be verified only up to a rather limited number of dimensions (less than 20). Nevertheless, in [12], two claims about the higher-dimensional statistics have been based on these two observations: First, that pairwise correlations may be sufficient to determine the full statistics of binary responses, and secondly, that the convergent scaling behavior in the log-log plot may indicate a transition towards strong order.

Using natural images instead of retinal ganglion cell data, we would like to verify to what extent the low-dimensional observations can be used to support these claims about the high-dimensional statistics [10]. To this end we study the same kind of extrapolation (Fig. 4 B) to higher dimensions (dashed lines) as in [12]. The difference between the entropy of the independent model and the multi-information yields the joint entropy of the respective distribution. If the extrapolation is taken seriously, this difference seems to vanish at the order of 50 dimensions suggesting that the joint entropy of the neural responses approaches zero at this size—say for $7 \times 7$ image patches (Fig. 4 C).

Though it was not taken literally, this point of 'freezing' has been pointed out in [12] as a critical network size at which a transition to strong order is to be expected. The meaning of this assertion, however, is not clear. First of all, the joint entropy of a distribution can never be smaller than the joint entropy of any of its marginals. Therefore, the joint entropy cannot decrease with increasing number of dimensions as the extrapolation would suggest (Fig. 4 C). Instead it would be necessary to ask more precisely how the growth rate of the joint entropy can be characterized and whether there is a critical number of dimensions at which the growth rate suddenly drops. In our study with natural images, visual inspection does not indicate anything special to happen at the 'critical patch size' of $7 \times 7$ pixels. Rather, for all patch sizes, the DG model yields dichotomized pink noise. In Fig. 5 (right) we show a sample from the DG model for $32 \times 32$ image patches (i.e. 1024 dimensions) which provides no indication for a particularly interesting change in the statistics towards strong order. The exact law according to which the multi-information grows with the number of dimensions for large $m$, however, is not easily assessed and remains to be explored.

Finally, we point out that the sufficiency of pairwise correlations at the level of $m = 16$ dimensions does not hold any more in the case of large $m$: the samples from the true distribution at the left hand side of Fig. 5 clearly show much more structure than the samples from the DG model (Fig. 5, right), indicating that pairwise correlations do not suffice to determine the full statistics of large image patches. Even if the match between the DG model and the Ising model may turn out to be less accurate in high dimensions, this would not affect our conclusion. Any mismatch would only introduce more order in the DG model than justified by pairwise correlations only.

## 5 Conclusion and Outlook

We proposed a new approach to maximum entropy modeling of binary variables, extending maximum entropy analysis to previously infeasible high dimensions: As both sampling and finding pa-

rameters is easy for the dichotomized Gaussian model, it overcomes the computational drawbacks of Monte-Carlo methods. We verified numerically that the empirical entropy of the DG model is comparable to that obtained with Gibbs sampling at least up to 20 dimensions. For practical purposes, the DG distribution can even be superior to the Gibbs sampler in terms of entropy maximization due to the lack of independence between consecutive samples in the Gibbs sampler.

Although the Ising model and the DG model are in principle different, the match between the two turns out to be surprisingly good for a large region of the parameter space. Currently, we are trying to determine where the close similarity between the Ising model and the DG model breaks down. In addition, we explore the possibility to use the dichotomized Gaussian distribution as a proposal density for Monte-Carlo methods such as importance sampling. As it is a very close approximation to the Ising model, we expect this combination to yield highly efficient sampling behaviour. In summary, by linking the DG model to the Ising model, we believe that maximum entropy modeling of multivariate binary random variables will become much more practical in the future.

We used the DG model to investigate the role of second-order correlations in the context of sensory coding of natural images. While for small image patches the DG model provided an excellent fit to the true distribution, we were able to show that this agreement breakes down in the case of larger image patches. Thus caution is required when extrapolating from low-dimensional measurements to higher-dimensional distributions because higher-order correlations may be invisible in low-dimensional marginal distributions. Nevertheless, the maximum entropy approach seems to be a promising tool for the analysis of correlated neural activities, and the DG model can facilitate its use significantly in practice.

### Acknowledgments

We thank Jakob Macke, Pierre Garrigues, and Greg Stephens for helpful comments and stimulating discussions, as well as Alexander Ecker and Andreas Hoenselaar for last minute advice. An implementation of the DG model in Matlab and R will be avaible at our website `http://www.kyb.tuebingen.mpg.de/bethgegroup/code/DGsampling`.

## Footnotes

[1] For integration, we used the mvncdf function of Matlab. For $m \geq 4$ this function employs Monte-Carlo integration.

## References

[1] D.H. Ackley, G.E. Hinton, and T.J. Sejnowski. A learning algorithm for boltzmann machines. *Cognitive Science*, 9:147–169, 1985.

[2] H.B. Barlow. Sensory mechanisms, the reduction of redundancy, and intelligence. In *The Mechanisation of Thought Processes*, pages 535–539, London: Her Majesty's Stationery Office, 1959.

[3] M. Bethge. Factorial coding of natural images: How effective are linear model in removing higher-order dependencies? *J. Opt. Soc. Am. A*, 23(6):1253–1268, June 2006.

[4] D.R. Cox and N. Wermuth. On some models for multivariate binary variables parallel in complexity with the multivariate gaussian distribution. *Biometrika*, 89:462–469, 2002.

[5] L.J. Emrich and M.R. Piedmonte. A method for generating high-dimensional multivariate binary variates. *The American Statistician*, 45(4):302–304, 1991.

[6] M. Huber. A bounding chain for swendsen-wang. *Random Structures & Algorithms*, 22:53–59, 2002.

[7] E.T. Jaynes. Where do we stand on maximum entropy inference. In R.D. Levine and M. Tribus, editors, *The Maximum Entropy Formalism*. MIT Press, Cambridge, MA, 1978.

[8] J. Linn. Divergence measures based on the shannon entropy. *IEEE Trans Inf Theory*, 37:145–151, 1991.

[9] D. J. C. MacKay. *Information Theory, Inference and Learning Algorithms*. Cambridge University Press, 2003.

[10] Sheila H Nirenberg and Jonathan D Victor. Analyzing the activity of large populations of neurons: how tractable is the problem? *Current Opinion in Neurobiology*, 17:397–400, August 2007.

[11] Karl Pearson. On a new method of determining correlation between a measured character a, and a character b, of which only the percentage of cases wherein b exceeds (or falls short of) a given intensity is recorded for each grade of a. *Biometrika*, 7:96–105, 1909.

[12] Elad Schneidman, Michael J Berry, Ronen Segev, and William Bialek. Weak pairwise correlations imply strongly correlated network states in a neural population. *Nature*, 440(7087):1007–1012, Apr 2006.

[13] J Shlens, JD Field, JL Gauthier, MI Grivich, D Petrusca, A Sher, AM Litke, and EJ Chichilnisky. The structure of multi-neuron firing patterns in primate retina. *J Neurosci*, 26(32):8254–8266, Aug 2006.

[14] G. Tkacik, E. Schneidman, M.J. Berry, and W. Bialek. Ising models for networks of real neurons. *arXiv:q-bio.NC/0611072*, 1:1–4, 2006.

